# An Analog Visual Pre-Processing Processor Employing Cyclic Line Access in Only-Nearest-Neighbor-Interconnects Architecture

**Yusuke Nakashita**
Department of Frontier Informatics
School of Frontier Sciences
The University of Tokyo
5-1-5 Kashiwanoha, Kashiwa-shi, Chiba
277-8561, Japan
yusuke@else.k.u-tokyo.ac.jp

**Yoshio Mita**
Department of Electrical Engineering
School of Engineering
The University of Tokyo
7-3-1 Hongo, Bunkyo-ku,Tokyo
113-8656, Japan.
mita@ee.t.u-tokyo.ac.jp

**Tadashi Shibata**
Department of Frontier Informatics
School of Frontier Sciences
The University of Tokyo
5-1-5 Kashiwanoha, Kashiwa-shi, Chiba
277-8561, Japan
shibata@ee.t.u-tokyo.ac.jp

## Abstract

An analog focal-plane processor having a $128 \times 128$ photodiode array has been developed for directional edge filtering. It can perform $4 \times 4$-pixel kernel convolution for entire pixels only with 256 steps of simple analog processing. Newly developed cyclic line access and row-parallel processing scheme in conjunction with the "only-nearest-neighbor interconnects" architecture has enabled a very simple implementation. A proof-of-concept chip was fabricated in a 0.35-$\mu$m 2-poly 3-metal CMOS technology and the edge filtering at a rate of 200 frames/sec. has been experimentally demonstrated.

## 1 Introduction

Directional edge detection in an input image is the most essential operation in early visual processing [1, 2]. Such spatial filtering operations are carried out by taking the convolution between a block of pixels and a weight matrix, requiring a number of multiply-and-accumulate operations. Since the convolution operation must be repeated pixel-by-pixel to scan the entire image, the computation is very expensive and software solutions are not compatible to real-time applications. Therefore, the hardware implementation of focal-plane parallel processing is highly demanded. However, there exists a hard problem which we call the *interconnects explosion* as illustrated in Fig. 1.

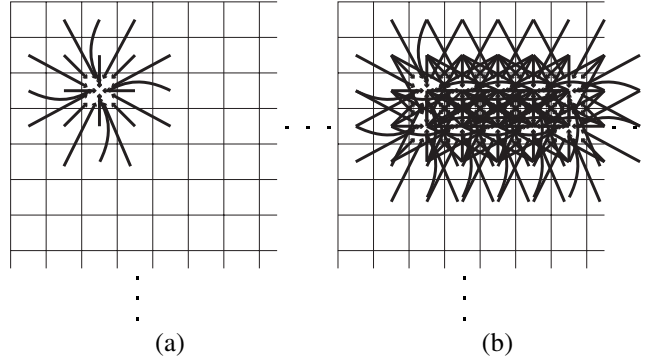

Figure 1: (a) Interconnects from nearest neighbor (N.N.) and second N.N. pixels to a single pixel at the center. (b) N.N. and second N.N. interconnects for pixels in the two rows, an illustrative example of *interconnecs explosion.*

In carrying out a filtering operation for one pixel, the luminance data must be gathered from the nearest-neighbor and second nearest-neighbor pixels. The interconnects necessary for this is illustrated in Fig. 1(a). If such wiring is formed for two rows of pixels, excessively high density overlapping interconnects are required. If we extend this to an entire chip, it is impossible to form the wiring even with the most advanced VLSI interconnects technology. Biology has solved the problem by real *3D-interconnects* structures. Since only two dimensional layouts are allowed with a limited number of stacks in VLSI technology, the *missing one dimension* is crucial. We must overcome the difficulty by introducing new architectures.

In order to achieve real-time performance in image filtering, a number of VLSI chips have been developed in both digital [3, 4] and analog [5, 6, 7] technologies. A flash-convolution processor [4] allows a single $5\times5$-pixel convolution operation in a single clock cycle by introducing a subtle memory access scheme. However, for an $N\times M$-pixel image, it takes $N\times M$ clock cycles to complete the processing. In the line-parallel processing scheme employed in [7], both row-parallel and column-parallel processing scan the target image several times and the entire filtering finishes in $O(N+M)$ steps. (A single step includes several clock cycles to control the analog processing.)

The purpose of this work is to present an analog focal-plane CMOS image sensor chip which carries out the directional edge filtering convolution for an $N\times M$-pixel image only in $M$ (or $N$) steps. In order to achieve an efficient processing, two key technologies have been introduced: "only-nearest-neighbor interconnects" architecture and "cyclic line access and row-parallel processing". The former was first developed in [8], and has enabled the convolution including second-nearest-neighbor luminance data only using nearest neighbor interconnects, thus greatly reducing the interconnect complexity. However, the fill factor was sacrificed due to the pixel parallel organization. The problem has been resolved in the present work by "cyclic line access and row-parallel processing." Namely, the processing elements are separated from the array of photo diodes and the "only-nearest-neighbor interconnects" architecture was realized as a separate module of row-parallel processing elements. The cyclic line access scheme first introduced in the present work has eliminated the redundant data readout operations from the photodiode array and has established a very efficient processing. As a result, it has become possible to complete the edge filtering for a $128\times128$ pixel image only in $128\times2$ steps. A proof-of-concept chip was fabricated in a 0.35-$\mu$m 2-poly 3-metal CMOS technology, and the edge detection at a rate of 200 frames/sec. has been experimentally demonstrated.

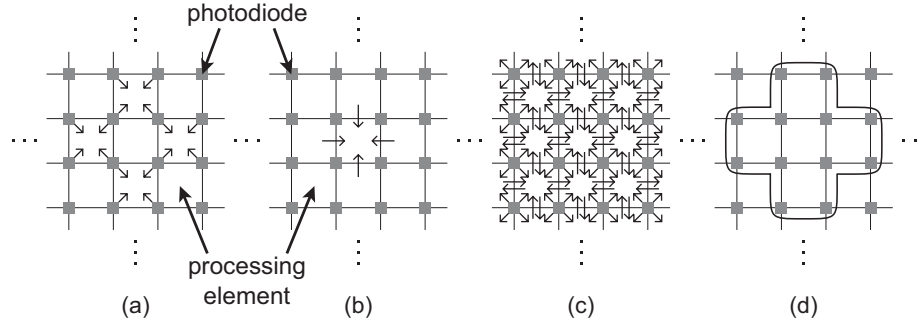

Figure 2: Edge filtering in the "only-nearest-neighbor interconnects" architecture: (a) first step; (b) second step; (c) all interconnects necessary for pixel parallel processing; (d) PD's involved in the convolution.

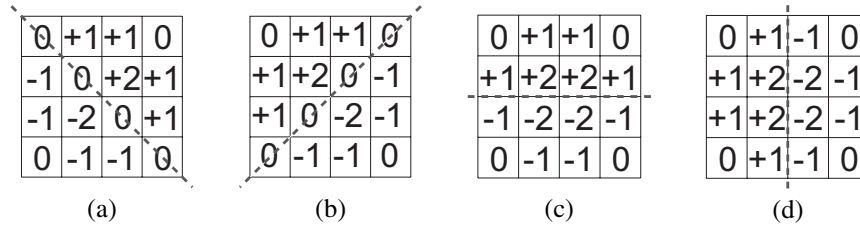

Figure 3: Edge filtering kernels realized in "only-nearest-neighbor interconnects" architecture: (a) $+45$ degree; (b) $-45$ degree; (c) horizontal; (d) vertical.

## 2   System Organization

The two key technologies employed in the present work are explained in the following.

### 2.1   "Only-Nearest-Neighbor Interconnects" Architecture

This architecture was first proposed in [8], and experimentally verified with small-scale test circuits ($7 \times 7$ processing elements without photodiodes). The key feature of the architecture is that photodiodes (PD's) are placed at four corners of each processing element (PE), and that the luminance data of each PD are shared by four PE's as shown in Fig. 2.

The edge filtering is carried out as explained below. First, as shown in Fig. 2 (a), pre-processing is carried out in each PE using the luminance data taken from four PD's located at its corners. Then, the result is transferred to the center PE as shown in Fig. 2 (b) and necessary computation is carried out. This accomplishes the filtering processing for one half of the entire pixels. Then the roles of pre-processing PE's and center PE's are interchanged and the same procedure follows to complete the processing for the rest of the pixels. The interconnects necessary for the entire parallel processing is shown in Fig. 2(c). In this manner, every PE can gather all data necessary for the processing from its nearest-neighbor and second nearest-neighbor pixels without complicated crossover interconnects. The kernels illustrated in Fig. 3 have been all realized in this architecture. The luminance data from 12 PD's enclosed in Fig. 2 (d) are utilized to detect the edge information at the center location.

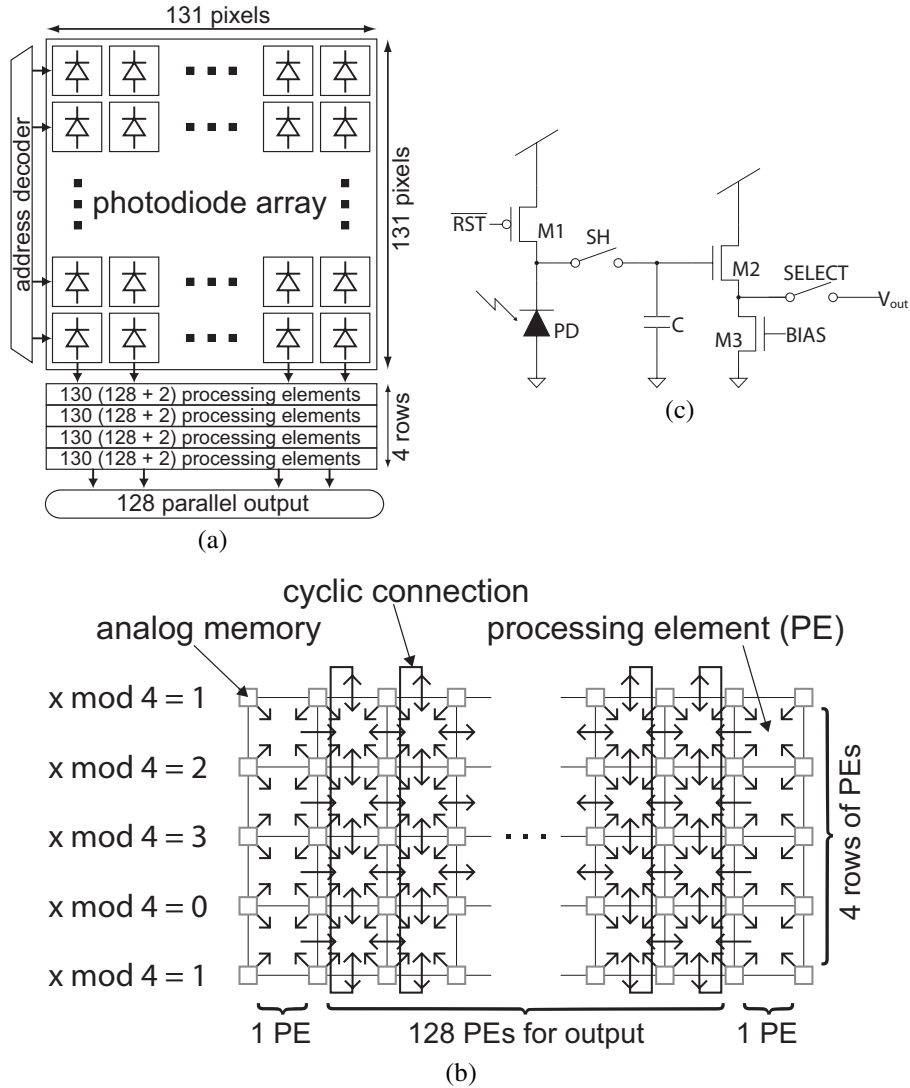

Figure 4: Block diagram of the chip (a), and organization of row-parallel processing module (b). $x$ in (b) represents the row number 1~131. (c) shows read out circuit of photodiode.

## 2.2 Cyclic Line Access and Row-Parallel Processing

A block diagram of the analog edge-filtering processor is given in Fig. 4 (a). It consists of an array of $131 \times 131$ photodiodes (PD's) and a module for row-parallel processing placed at the bottom of the PD array. Figure 4(b) illustrates the organization of the row processing module, which is composed of four rows of 130 PE's and five rows of 131 analog memory cells that temporarily store the luminance data read out from the PD array. It should be noted that only three rows of PE's and four rows of PD's are sufficient to carry out a single-row processing as explained in reference to Fig. 2(d). However, one extra row of PE's and one extra row of analog memories for PD data storage were included in the row-parallel processing module. This is essential to carry out a seamless data read out from the PD array and computation without analog data shift within the processing module. The chip yields

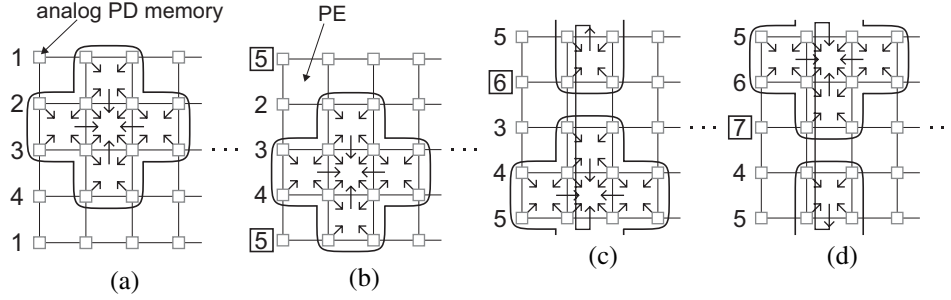

Figure 5: "Cyclic-line access and row-parallel processing" scheme.

the kernel convolution results for one of the rows in the PD array as 128 parallel outputs.

Now, the operation of the row-parallel processing module is explained with reference to Fig. 4 (b) and Fig. 5. In order to carry out the convolution for the data in Row 1~4, the PD data are temporarily stored in the analog memory array as shown in Fig. 5 (a). Imporatant to note is that the data from Row 1 are duplicated at the bottom. The convolution operation proceeds using the upper four rows of data as explained in Fig. 5 (a). In the next step, the data from Row 5 are overwritten to the sites of Row 1 data as shown in Fig. 5 (b). The operation proceeds using the lower four rows of data and the second set of outputs is produced. In the third step, the data from Row 6 is overwritten to the sites of Row 2 data (Fig. 5 (c)), and the convolution is taken using the data in the enclosure. Although a part of the data (top two rows) are separated from the rest, the topology of the hardware computation is identical to that explained in Fig. 5 (a). This is because the same set of data is stored in both top and bottom PD memories and the top and bottom PE's are connected by "cyclic connection" as illustrated in Fig. 4 (b). By introducing such one extra row of PD memories and one extra row of PE's with cyclic interconnections, row-parallel processing can be seamlessly performed with only a single-row PD data set download at each step.

## 3 Circuit Configurations

In this architecture, we need only two arithmetic operations, i.e., the sum of four inputs and the subtraction.

Figure 6(a) shows the adder circuit using the multiple-input floating-gate source follower [9]. The substrate of $M1$ is connected to the source to avoid the body effect. The transistor $M2$ operates as a current source for fast output voltage stabilization as well as to achieve good linearity. Due to the charge redistribution in the floating gate, the average of the four input voltages appears at the output as

$$V_{\text{out}} = \frac{V_1 + V_2 + V_3 + V_4}{4} + |V_{\text{th}}|,$$

where $V_{\text{th}}$ represents the threshold voltage of $M1$. Here, the four coupling capacitors connected to the floating gate of $M1$ are identical and the capacitance coupling between the floating gate and the ground was assumed to be 0 for simplicity. The electrical charge in the floating gate is initialized periodically using the reset switch ($M3$). The coupling capacitors themselves are also utilized as temporary memories for the PD data read out from the PD array.

Figure 6(b) shows the subtraction circuit, where the same source follower was used. When SW1 and SW2 are turned on, and SW3 is turned off, the following voltage difference is

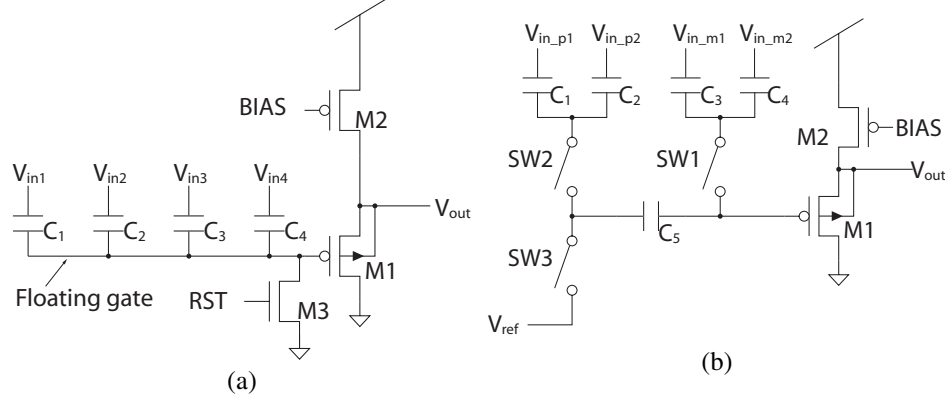

Figure 6: Adder circuit (a) and subtraction circuit (b) using floating-gate MOS technology.

developed across the capacitor $C_5$:

$$\frac{(V_{\text{in\_p1}} + V_{\text{in\_p2}}) - (V_{\text{in\_m1}} + V_{\text{in\_m2}})}{4}.$$

Then, SW1 and SW2 are turned off, and SW3 is turned on. As a result, the output voltage $V_{\text{out}}$ becomes

$$V_{\text{out}} = \frac{(V_{\text{in\_p1}} + V_{\text{in\_p2}}) - (V_{\text{in\_m1}} + V_{\text{in\_m2}})}{4} + V_{\text{ref}} + |V_{\text{th}}|,$$

where $V_{\text{th}}$ represents the threshold voltage of $M1$.

## 4 Experimental Results

A proof-of-concept chip was designed and fabricated in a 0.35-$\mu$m 2-poly 3-metal CMOS technology. Figure 7 shows the photomicrograph of the chip, and the chip specifications are given in Table 1. Since the pitch of a single PE unit is larger than the pitch of the PD array, 130 PE units are laid out as two separate L-shaped blocks at the periphery of the PD array as seen in the chip photomicrograph. Successful operation of the chip was experimently verified.

An example is shown in Fig. 8, where the experimental results for $-45$-degree edge filtering are demonstrated. Since the thresholding circuitry was not implemented in the present chip, only the convolution results are shown. 128 parallel outputs from the test chip were multiplexed for observation using the external multiplexers mounted on a milled printed circuit board. The vertical stripes observed in the result are due to the resistance variation in the external interconnects poorly produced on the milled printed circuit board.

It was experimentally confirmed the chip operates at 1000 frames/sec. However, the operation is limited by the integration time of PD's and typical motion images are processed at about 200 frames/sec. The power dissipation in the PE's was 25 mW and that in the PD array was 40mW.

## 5 Conclusions

An analog edge-filtering processor has been developed based on the two key technologies: "only-nearest-neighbor interconnects" architecture and "cyclic line access and row-parallel

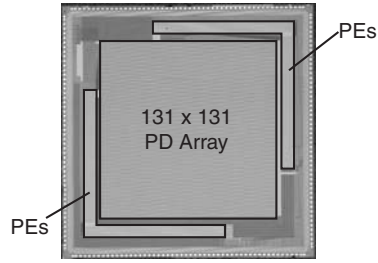

Figure 7: Chip photomicrograph.

| Table 1: Chip Specifications. | |
|---|---|
| Process Technology | 0.35 $\mu$m CMOS, 2-Poly, 3-Metal |
| Die Size | 9.8 mm x 9.8 mm |
| Voltage Supply | 3.3 V |
| Operating Frequency | 50M Hz |
| Power Dissipation | 25 mW (PE Array) |
| PE Operation | 1000 Frames/secl |
| Typical Frame Ratel | 200 Frames / sec (limited by PD integration time) |

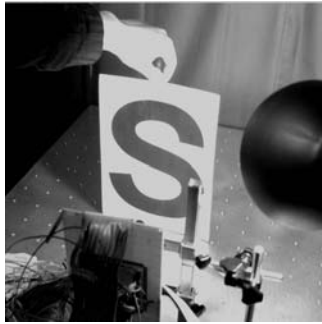
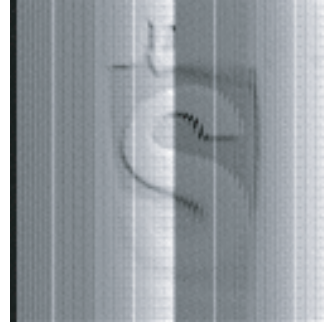

(a)      (b)

Figure 8: Experimental set up (a), and measurement results of $-45$ degree edge filtering convolution (b).

processing". As a result, the convolution operation involving second nearest-neighbor pixel data for an $N \times N$-pixel image can be performed only in $2N$ steps. The edge filtering operation for $128 \times 128$-pixel images at 200 frames/sec. has been experimentally demonstrated. The chip meets the requirement of low-power and real-time-response applications.

## 6  Acknowledgments

The VLSI chip in this study was fabricated in the chip fabrication program of VLSI Design and Education Center (VDEC), the University of Tokyo in collaboration with Rohm Corporation and Toppan Printing Corporation. The work is partially supported by the Ministry of Education, Science, Sports, and Culture under Grant-in-Aid for Scientific Research (No. 14205043).

## References

[1] D. H. Hubel and T. N. Wiesel, "Receptive fields of single neurons in the cat's striate cortex," *Journal of Physiology*, vol. 148, pp. 574-591, 1959.

[2] M. Yagi and T. Shibata, "An image representation algorithm compatible with neural-associative-processor-based hardware recognition systems," *IEEE Trans. Neural Networks*, vol. 14(5), pp. 1144-1161, 2003.

[3] J. C. Gealow and C. G. Sodini, "A pixel parallel-processor using logic pitch-matched to dynamic memory," *IEEE J. Solid-State Circuits*, vol. 34, pp. 831-839, 1999.

[4] K. Ito, M. Ogawa and T. Shibata, "A variable-kernel flash-convolution image filtering processor," *Dig. Tech. Papers of Int. Solid-State Circuits Conf.*, pp. 470-471, 2003.

[5] L. D. McIlrath, "A CCD/CMOS focal plane array edge detection processor implementing the multiscale veto algorithm," *IEEE J. Solid-State Circuits*, vol. 31(9), pp. 1239-1247, 1996.

[6] R. Etiene-Cummings, Z. K. Kalayjian and D. Cai, "A programmable focal plane MIMD image processor chip," *IEEE J. Solid-State Circuits*, vol. 36(1), pp. 64-73, 2001.

[7] T. Taguchi, M. Ogawa and T. Shibata, "An Analog Image Processing LSI Employing Scanning Line Parallel Processing," *Proc. 29th European Solid-Sate Circuits Conference (ESSCIRC 2003)*, pp. 65-68, 2003.

[8] Y. Nakashita, Y. Mita and T. Shibata, "An Analog Edge-Filtering Processor Employing Only-Nearest-Neighbor Interconnects," *Ext. Abstracts of the International Conference on Solid State Devices and Materials (SSDM '04)*, pp. 356-357, 2004.

[9] T. Shibata and T. Ohmi, "A Functional MOS Transistor Featuring Gate-Level Weighted Sum and Threshold Operations," *IEEE Trans. Electron Devices*, vol. 39(6), pp. 1444-1455, 1992.
